# Kirchoff Law Markov Fields for Analog Circuit Design

Richard M. Golden *
RMG Consulting Inc.
2000 Fresno Road, Plano, Texas 75074
*RMGCONSULT@AOL.COM*,
*www.neural-network.com*

## Abstract

Three contributions to developing an algorithm for assisting engineers in designing analog circuits are provided in this paper. First, a method for representing highly nonlinear and non-continuous analog circuits using Kirchoff current law potential functions within the context of a Markov field is described. Second, a relatively efficient algorithm for optimizing the Markov field objective function is briefly described and the convergence proof is briefly sketched. And third, empirical results illustrating the strengths and limitations of the approach are provided within the context of a JFET transistor design problem. The proposed algorithm generated a set of circuit components for the JFET circuit model that accurately generated the desired characteristic curves.

## 1 Analog circuit design using Markov random fields

### 1.1 Markov random field models

A Markov random field (MRF) is a generalization of the concept of a Markov chain. In a Markov field one begins with a set of random variables and a *neighborhood relation* which is represented by a graph. Each random variable will be assumed in this paper to be a discrete random variable which takes on one of a finite number of possible values. Each node of the graph indexes a specific random variable. A link from the $j$th node to the $i$th node indicates that the conditional probability distribution of the $i$th random variable in the field is functionally dependent upon the $j$th random variable. That is, random variable $j$ is a *neighbor* of random variable $i$. The only restriction upon the definition of a Markov field (i.e., the *positivity condition*) is that the probability of every realization of the field is strictly positive. The essential idea behind Markov field design is that one specifies a potential (energy) function for every clique in the neighborhood graph such that the subset of random variables associated with that clique obtain their optimal values when that clique's potential function obtains its minimal value (for reviews see [1]-[2]).

---

* Associate Professor at University of Texas at Dallas (*www.utdallas.edu/~golden*)

Markov random field models provide a convenient mechanism for probabilistically representing and optimally combining combinations of local constraints.

## 1.2   Analog circuit design using SPICE

In some mixed signal ASIC (Application Specific Integrated Circuit) design problems, most of the circuit design specifications are well known but the introduction of a single constraint (e.g., an increase in substrate noise) could result in a major redesign of an entire circuit. The industry standard tool for aiding engineers in solving analog circuit design problems is SPICE which is a software environment for simulation of large scale electronic circuits. SPICE does have special optimization options for fitting circuit parameters to desired input-output characteristics but typically such constraints are too weak for SPICE to solve analog circuit design problems with large numbers of free parameters (see [3] for an introduction to SPICE). Another difficulty with using SPICE is that it does not provide a global confidence factor for indicating its confidence in a generated design or local confidence factors for determining the locations of "weak points" in the automatically generated circuit design solution.

## 1.3   Markov field approaches to analog circuit design

In this paper, an approach for solving real-world analog circuit design problems using an appropriately constructed Markov random is proposed which will be referred to as MRFSPICE. Not only are desired input-output characteristics directly incorporated into the construction of the potential functions for the Markov field but additional constraints based upon Kirchoff's current law are directly incorporated into the field. This approach thus differs from the classic SPICE methodology because Kirchoff current law constraints are explicitly incorporated into an objective function which is minimized by the "optimal design". This approach also differs from previous Markov field approaches (i.e., the "Harmony" neural network model [4] and the "Brain-State-in-a-Box" neural network model [5]) designed to qualitatively model human understanding of electronic circuit behavior since those approaches used pair-wise correlational (quadratic) potential functions as opposed to the highly nonlinear potential functions that will be used in the approach described in this paper.

## 1.4   Key contributions

This paper thus makes three important contributions to the application of Markov random fields to the analog circuit design problem. First, a method for representing highly nonlinear and non-continuous analog circuits using Kirchoff current law potential functions within the context of a Markov field is described. Second, a relatively efficient algorithm for optimizing the Markov field objective function is briefly described and the convergence proof is briefly sketched. And third, empirical results illustrating the strengths and limitations of the approach is provided within the context of a JFET transistor design problem.

## 2   Modeling assumptions and algorithms

### 2.1   Probabilistic modeling assumptions

A given circuit circuit design problem consists of a number of design decision variables. Denote those design decision variables by the discrete random variables

$\tilde{x}_1, \ldots, \tilde{x}_d$. Let the MRF be denoted by the set $\bar{\mathbf{x}} = [\tilde{x}_1, \ldots, \tilde{x}_d]$ so that a realization of $\bar{\mathbf{x}}$ is the $d$-dimensional real vector $\mathbf{x}$. A realization of $\bar{\mathbf{x}}$ is referred to as a *circuit design solution.*

Let the joint (global) probability mass function for $\bar{\mathbf{x}}$ be denoted by $p_G$. It is assumed that $p_G(\mathbf{x}) > p_G(\mathbf{y})$ if and only if the circuit design solution $\mathbf{x}$ is preferred to the circuit design solution $\mathbf{y}$. Thus, $p_G(\mathbf{x})$ specifies a type of probabilistic fuzzy measure [1].

For example, the random variable $\tilde{x}_i$ might refer to a design decision concerning the choice of a particular value for a capacitor $C_{14}$. From previous experience, it is expected that the value of $C_{14}$ may be usually constrained without serious difficulties to one of ten possible values:

$$0.1\mu F, 0.2\mu F, 0.3\mu F, 0.4\mu F, 0.5\mu F, 0.6\mu F, 0.7\mu F, 0.8\mu F, 0.9\mu F, \quad or \quad 1\mu F.$$

Thus, $k_i = 10$ in this example. By limiting the choice of $C_{14}$ to a small number of finite values, this permits the *introduction of design expertise hints* directly into the problem formulation without making strong committments to the ultimate choice of the value of capacitor $C_{14}$. Other examples of design decision variable values include: resistor values, inductor values, transistor types, diode types, or even fundamentally different circuit topologies.

The problem that is now considered will be to assign design preference probabilities in a meaningful way to alternative design solutions. The strategy for doing this will be based upon constructing $p_G$ with the property that if $p_G(\mathbf{x}) > p_G(\mathbf{y})$, then circuit design solution $\mathbf{x}$ exhibits the requisite operating characteristics with respect to a set of $M$ "test circuits" more effectively than circuit design solution $\mathbf{y}$. An optimal analog circuit design solution $\mathbf{x}^*$ then may be defined as a global maximum of $p_G$. The specific details of this strategy for constructing $p_G$ are now discussed by first carefully defining the concept of a "test circuit".

Let $\mathcal{V} = \{0, 1, 2, \ldots, m\}$ be a finite set of integers (i.e., the unique "terminals" in the test circuit) which index a set of $m$ complex numbers, $v_0, v_1, v_2, \ldots, v_m$ which will be referred to as *voltages*. The magnitude of $v_k$ indicates the voltage magnitude while the angle of $v_k$ indicates the voltage phase shift. By convention the *ground voltage*, $v_0$, is always assigned the value of 0. Let $d \in \mathcal{V} \times \mathcal{V}$ (i.e., an ordered pair of elements in $\mathcal{V}$). A *circuit component current source* is defined with respect to $\mathcal{V}$ by a complex-valued function $i_{a,b}$ whose value is typically functionally dependent upon $v_a$ and $v_b$ but may also be functionally dependent upon other voltages and circuit component current sources associated with $\mathcal{V}$.

For example, a "resistor" circuit component current source would be modeled by choosing $i_{a,b} = (v_b - v_a)/R$ where $R$ is the resistance in ohms of some resistor, $v_b$ is the voltage observed on one terminal of the resistor, and $v_a$ is the voltage observed on the other terminal of the resistor. The quantity $i_{a,b}$ is the current flowing through the resistor from terminal $a$ to terminal $b$. Similarly, a "capacitor" circuit component current source would be modeled by choosing $i_{a,b} = (v_b - v_a)/[2\pi j f]$ where $j = \sqrt{-1}$ and $f$ is the frequency in $Hz$ of the test circuit. A "frequency specific voltage controlled current source" circuit component current source may be modeled by making $i_{a,b}$ functionally dependent upon some subset of voltages in the test circuit. See [6] for additional details regarding the use of complex arithmetic for analog circuit analysis and design.

An important design constraint is that Kirchoff's current law should be satisfied at every voltage node. Kirchoff's current law states that the sum of the currents entering a voltage node must be equal to zero [6]. We will now show how this physical law can be directly embodied as a system of nonlinear constraints on the

behavior of the MRF.

We say that the $k$th voltage node in test circuit $q$ is *clamped* if the voltage $v_k$ is known. For example, node $k$ in circuit $q$ might be directly grounded, node $k$ might be directly connected to a grounded voltage source, or the voltage at node $k$, $v_k$, might be a desired known target voltage.

If voltage node $k$ in test circuit $q$ is clamped, then Kirchoff's current law at voltage node $k$ in circuit $q$ is simply assumed to be satisfied which, in turn, implies that the *voltage potential function* $\Phi_{q,k} = 0$.

Now suppose that voltage node $k$ in test circuit $q$ is not clamped. This means that the voltage at node $k$ must be estimated. If there are no controlled current sources in the test circuit (i.e., only passive devices), then the values of the voltages at the unclamped nodes in the circuit can be calculated by solving a system of linear equations where the current choice of circuit component values are treated as constants. In the more general case where controlled current sources exist in the test circuit, then an approximate iterative gradient descent algorithm (such as the algorithm used by SPICE) is used to obtain improved estimates of the voltages of the unclamped nodes. The iterative algorithm is always run for a fixed number of iterations.

Now the value of $\Phi_{q,k}$ must be computed. The current entering node $k$ via arc $j$ in test circuit $q$ is denoted by the two-dimensional real vector $\mathbf{I}_{k,j}^q$ whose first component is the real part of the complex current and whose second component is the imaginary part.

The average current entering node $k$ in test circuit $q$ is given by the formula:

$$\bar{\mathbf{I}}_k^q = (1/n_k) \sum_{j=1}^{n_k} \mathbf{I}_{k,j}^q.$$

Design circuit components (e.g., resistors, capacitors, diodes, etc.) which minimize $\bar{\mathbf{I}}_k^q$ will satisfy Kirchoff's current law at node $k$ in test circuit $q$. However, the measure $\bar{\mathbf{I}}_k^q$ is an not entirely adequate indicator of the degree to which Kirchoff's current law is satisfied since $\bar{\mathbf{I}}_k^q$ may be small in magnitude not necessarily because Kirchoff's current law is satisfied but simply because all currents entering node $k$ are small in magnitude. To compensate for this problem, a normalized current signal magnitude to current signal variability ratio is minimized at node $k$ in test circuit $q$. This ratio decreases in magnitude if $\bar{\mathbf{I}}_k^q$ has a magnitude which is small relative to the magnitude of individual currents entering node $k$ in test circuit $q$.

The voltage potential function, $\Phi_{q,k}$, for voltage node $k$ in test circuit $q$ is now formally defined as follows. Let

$$\mathbf{Q}_{k,q} = (1/n_k) \sum_{j=1}^{n_k} (\mathbf{I}_{k,j}^q - \bar{\mathbf{I}}_k^q)(\mathbf{I}_{k,j}^q - \bar{\mathbf{I}}_k^q)^T.$$

Let $\lambda_1, \ldots, \lambda_u$ be those eigenvalues of $\mathbf{Q}_{k,q}$ whose values are strictly greater than some small positive number $\epsilon$. Let $\mathbf{e}_i$ be the eigenvector associated with eigenvalue $\lambda_i$. Define

$$\mathbf{Q}_{k,q}^{-1} = \sum_{j=1}^{u} (1/\lambda_j) \mathbf{e}_j \mathbf{e}_j^T.$$

Thus, if $\mathbf{Q}_{k,q}$ has all positive eigenvalues, then $\mathbf{Q}_{k,q}$ is simply the matrix inverse of $\mathbf{Q}_{k,q}^{-1}$. Using this notation, the voltage potential function for the unclamped voltage

node $k$ in test circuit $q$ may be expressed by the formula:

$$\Phi_{q,k} = [\bar{\mathbf{I}}_k^q]^T \mathbf{Q}^{-1} \bar{\mathbf{I}}_k^q.$$

Now define the global probability or "global preference" of a particular design configuration by the formula:

$$p_G(\mathbf{x}) = (1/Z)exp(-U(\mathbf{x})) \tag{1}$$

where $U = (1/N) \sum_q \sum_k \Phi_{q,k}$ and where $N$ is the total number of voltage nodes across all test circuits. The most preferred (i.e., "most probable") design are the design circuit components that maximize $p_G$. Note that probabilities have been assigned such that circuit configurations which are less consistent with Kirchoff's current law are considered "less probable" (i.e., "less preferred").

Because the normalization constant $Z$ in (1) is computationally intractable to compute, it is helpful to define the easily computable *circuit confidence factor, CCF*, given by the formula: $CCF(\mathbf{x}) = exp(-U(\mathbf{x})) = Zp_G(\mathbf{x})$. Note that the global probability $p$ is directly proportional to the $CCF$. Since $U$ is always non-negative and complete satisfaction of Kirchoff's current laws corresponds to the case where $U = 0$, it follows that $CCF(\mathbf{x})$ has a lower bound of 0 (indicating "no subjective confidence" in the design solution $\mathbf{x}$) and an upper bound of 1 (indicating "absolute subjective confidence" in the design solution $\mathbf{x}$).

In addition, local conditional probabilities of the form

$$p_i = p(\tilde{x}_i = x_i | x_1, \ldots, x_{i-1}, x_{i+1}, \ldots, x_d)$$

can be computed using the formula:

$$p_i = \frac{p_G(x_1, \ldots, x_{i-1}, x_i, x_{i+1}, \ldots, x_d)}{\sum_k p_G(x_1, \ldots, x_{i-1}, x_k, x_{i+1}, \ldots, x_d)}.$$

Such local conditional probabilities are helpful for explicitly computing the probability or "preference" for selecting one design circuit component value given a subset of other design component values have been accepted. Remember that probability (i.e., "preference") is essentially a measure of the degree to which the chosen design components and pre-specified operating characteristic voltage versus frequency curves of the circuit satisfy Kirchoff's current laws.

## 2.2 MRFSPICE algorithm

The MRFSPICE algorithm is a combination of the Metropolis and Besag's ICM (Iterated Conditional Modes) algorithms [1]-[2]. The stochastic Metropolis algorithm (with temperature parameter set equal to one) is used to sample from $p(\mathbf{x})$. As each design solution is generated, the CCF for that design solution is computed and the design solution with the best CCF is kept as an initial design solution guess $\mathbf{x}_0$. Next, the deterministic ICM algorithm is then initialized with $\mathbf{x}_0$ and the ICM algorithm is applied until an equilibrium point is reached.

A simulated annealing method involving decreasing the temperature parameter according to a logarithmic cooling schedule in Step 1 through Step 5 could easily be used to guarantee convergence in distribution to a uniform distribution over the global maxima of $p_G$ (i.e., convergence to an optimal solution) [1]-[2]. However, for the test problems considered thus far, equally effective results have been obtained by using the above fast heuristic algorithm which is guaranteed to converge to a local maximum as opposed to a global maximum. It is proposed that in situations where the convergence rate is slow or the local maximum generated by MRFSPICE is a

poor design solution with low CCF, that appropriate local conditional probabilites be computed and provided as feedback to a human design engineer. The human design engineer can then make direct alterations to the sample space of $p_G$ (i.e., the domain of *CCF*) in order to appropriately simply the search space. Finally, the ICM algorithm can be easily viewed as an artificial neural network algorithm and in fact is a generalization of the classic Hopfield (1982) model as noted in [1].

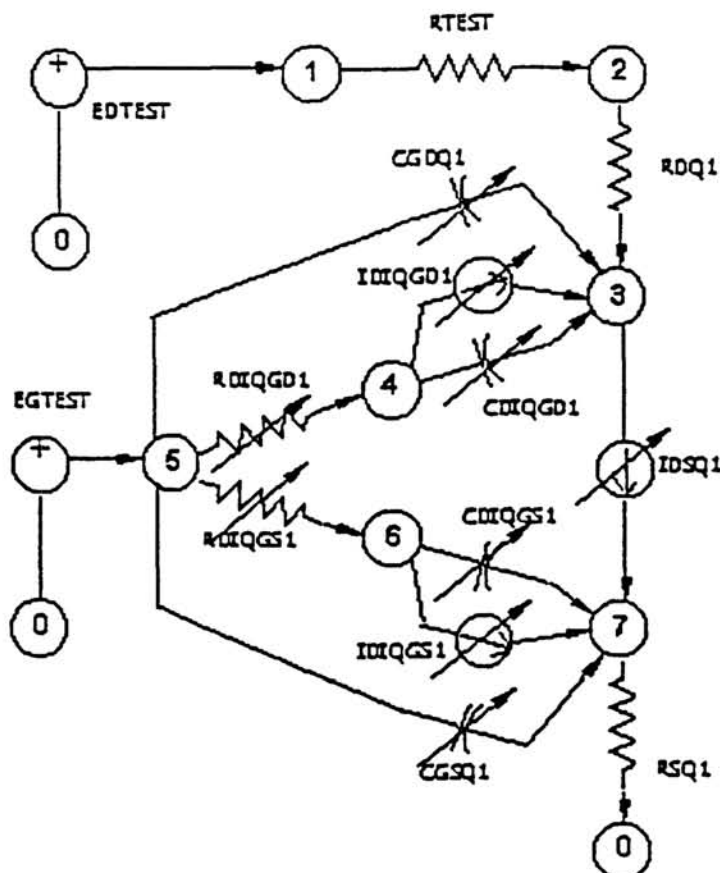

Figure 1: As external input voltage generator EGTEST and external supply voltage EDTEST are varied, current IRTEST flowing through external resistor RTEST is measured.

## 3    JFET design problem

In this design problem, specific combinations of free parameters for a macroequivalent JFET transistor model were selected on the basis of a given set of characteristic curves specifying how the drain to source current of the JFET varied as a function of the gate voltage and drain voltage at $0Hz$ and $1MHz$. Specifically, a JFET transistor model was simulated using the classic Shichman and Hodges (1968) large-signal n-channel JFET model as described by Vladimirescu [3] (pp. 96-100). The circuit diagram of this transistor model is shown in Figure 1. The only components in the circuit diagram which are not part of the JFET transistor model are the external voltage generators EDTEST and EGTEST, and external resistor RTEST. The specific functions which describe how IDIQGD1, CDIQGD1, RDIQGD1, IDIQGS1, CDIQGS1, RDIQGS1, CGDQ1, and CGSQ1 change as a function of EGTEST and the current IRTEST (which flows through RTEST) are too long and complex to be

presented here (for more details see [3] pp. 96-100).

Five design decision variables were defined. The first design decision variable, XDIQGS1, specified a set of parameter values for the large signal gate to source diode model portion of the JFET model. There were 20 possible choices for the value of XDIQGS1. Similarly, the second design decision variable, XDIQGD1, had 20 possible values and specified a set of parameter values for the large signal gate to drain diode model portion of the JFET model. The third design decision variable was XQ1 which also had 20 possible values were each value specified a set of choices for JFET-type specific parameters. The fourth and fifth design decision variables were the resistors RSQ1 and RSD1 each of which could take on one of 15 possible values.

The results of the JFET design problem are shown in Table 1. The phase angle for IRTEST at $1MHz$ was specified to be approximately 10 degrees, while the observed phase angle for IRTEST ranged from 7 to 9 degrees. The computing time was approximately $2 - 4$ hours using unoptimized prototype MATLAB code on a 200 MHZ Pentium Processor. The close agreement between the desired and actual results suggests further research in this area would be highly rewarding.

Table 1: Evaluation of MRFSPICE-generated JFET design

| EGTEST | EDTEST | IRTEST @ DC (ma) (desired/actual) | IRTEST @ 1MHZ (ma) (desired/actual) |
|---|---|---|---|
| 0 | 1.5 | 1.47/1.50 | 1.19/1.21 |
| 0 | 2.0 | 1.96/1.99 | 1.60/1.62 |
| 0 | 3.0 | 2.94/2.99 | 2.43/2.43 |
| -0.5 | 1.5 | 1.47/1.50 | 1.07/1.11 |
| -0.5 | 2.0 | 1.96/2.00 | 1.49/1.52 |
| -0.5 | 3.0 | 2.95/2.99 | 2.34/2.35 |
| -1.0 | 1.5 | 1.48/1.50 | 0.96/1.02 |
| -1.0 | 2.0 | 1.97/2.00 | 1.39/1.44 |
| -1.0 | 3.0 | 2.96/3.00 | 2.27/2.29 |

## Acknowledgments

This research was funded by Texas Instruments Inc. through the direct efforts of Kerry Hanson. Both Kerry Hanson and Ralph Golden provided numerous key insights and knowledge substantially improving this project's quality.

## References

[1] Golden, R. M. (1996) *Mathematical methods for neural network analysis and design.* Cambridge: MIT Press.

[2] Winkler, G. (1995) *Image analysis, random fields, and dynamic Monte Carlo methods: A mathematical introduction.* New York: Springer-Verlag.

[3] Vladimirescu, A. (1994) *The SPICE book.* New York: Wiley.

[4] Smolensky, P. (1986). Information processing in dynamical systems: Foundations of Harmony theory. In D. E. Rumelhart and J. L. McClelland (eds.), *Parallel distributed processing. Volume 1: Foundations*, pp. 194-281. Cambridge: MIT Press.

[5] Anderson, J. A. (1995). *An introduction to neural networks.* Cambridge: MIT Press.

[6] Skilling, H. (1959) *Electrical engineering circuits.* New York: Wiley.